# Online Linear Regression and Its Application to Model-Based Reinforcement Learning

**Alexander L. Strehl**[*]
Yahoo! Research
New York, NY
strehl@yahoo-inc.com

**Michael L. Littman**
Department of Computer Science
Rutgers University
Piscataway, NJ USA
mlittman@cs.rutgers.edu

## Abstract

We provide a provably efficient algorithm for learning Markov Decision Processes (MDPs) with continuous state and action spaces in the online setting. Specifically, we take a model-based approach and show that a special type of online linear regression allows us to learn MDPs with (possibly kernalized) linearly parameterized dynamics. This result builds on Kearns and Singh's work that provides a provably efficient algorithm for finite state MDPs. Our approach is not restricted to the linear setting, and is applicable to other classes of continuous MDPs.

## Introduction

Current reinforcement-learning (RL) techniques hold great promise for creating a general type of artificial intelligence (AI), specifically autonomous (software) agents that learn difficult tasks with limited feedback (Sutton & Barto, 1998). Applied RL has been very successful, producing world-class computer backgammon players (Tesauro, 1994) and model helicopter flyers (Ng et al., 2003). Many applications of RL, including the two above, utilize *supervised-learning techniques* for the purpose of *generalization*. Such techniques enable an agent to act intelligently in new situations by learning from past experience in different but similar situations.

Provably efficient RL for *finite state and action spaces* is accomplished by Kearns and Singh (2002) and hugely contributes to our understanding of the relationship between *exploration* and *sequential decision making*. The achievement of the current paper is to provide an efficient RL algorithm that learns in Markov Decision Processes (MDPs) with continuous state and action spaces. We prove that it learns linearly-parameterized MDPs, a model introduced by Abbeel and Ng (2005), with *sample (or experience) complexity* that grows only polynomially with the number of state space dimensions.

Our new RL algorithm utilizes a special linear regresser, based on least-squares regression, whose analysis may be of interest to the online learning and statistics communities. Although our primary result is for linearly-parameterized MDPs, our technique is applicable to other classes of continuous MDPs and our framework is developed specifically with such future applications in mind. The linear dynamics case should be viewed as only an interesting example of our approach, which makes substantial progress in the goal of understanding the relationship between exploration and generalization in RL.

An outline of the paper follows. In Section 1, we discuss online linear regression and pose a new online learning framework that requires an algorithm to not only provide predictions for new data points but also provide formal guarantees about its predictions. We also develop a specific algorithm and prove that it solves the problem. In Section 2, using the algorithm and result from the first section, we develop a provably efficient RL algorithm. Finally, we conclude with future work.

---

[*]Some of the work presented here was conducted while the author was at Rutgers University.

# 1  Online Linear Regression

Linear Regression (LR) is a well-known and tremendously powerful technique for prediction of the value of a variable (called the response or output) given the value of another variable (called the explanatory or input). Suppose we are given some data consisting of input-output pairs: $(x_1, y_1), (x_2, y_2), \ldots, (x_m, y_m)$, where $x_i \in \mathbb{R}^n$ and $y_i \in \mathbb{R}$ for $i = 1, \ldots, m$. Further, suppose that the data satisfies a linear relationship, that is $y_i \approx \theta^T x_i \ \forall i \in \{1, \ldots, m\}$, where $\theta \in \mathbb{R}^n$ is an $n$-dimensional parameter vector. When a new input $x$ arrives, we would like to make a prediction of the corresponding output by estimating $\theta$ from our data. A standard approach is to approximate $\theta$ with the *least-squares estimator* $\hat{\theta}$ defined by $\hat{\theta} = (X^T X)^{-1} X^T y$, where $X \in \mathbb{R}^{m \times n}$ is a matrix whose $i$th row consists of the $i$th input $x_i^T$ and $y \in \mathbb{R}^n$ is a vector whose $i$th component is the $i$th output $y_i$.

Although there are many analyses of the linear regression problem, none is quite right for an application to model-based reinforcement learning (MBRL). In particular, in MBRL, we cannot assume that $X$ is fixed ahead of time and we require more than just a prediction of $\theta$ but knowledge about whether this prediction is sufficiently accurate. A robust learning agent must not only infer an approximate model of its environment but also maintain an idea about the accuracy of the parameters of this model. Without such meta-knowledge, it would be difficult to determine when to explore (or when to trust the model) and how to explore (to improve the model). We coined the term KWIK ("know what it knows") for algorithms that have this special property. With this idea in mind, we present the following online learning problem related to linear regression. Let $||v||$ denote the Euclidean norm of a vector $v$ and let $\text{Var}\,[X]$ denote the variance of a random variable $X$.

**Definition 1** (**KWIK Linear Regression Problem or** KLRP) *On every timestep $t = 1, 2, \ldots$ an input vector $x_t \in \mathbb{R}^n satisfying ||x_t|| \leq 1$ and output number $y_t \in [-1, 1]$ is provided. The input $x_t$ may be chosen in any way that depends on the previous inputs and outputs $(x_1, y_1), \ldots, (x_t, y_t)$. The output $y_t$ is chosen probabilistically from a distribution that depends only on $x_t$ and satisfies $E[y_t] = \theta^T x_t$ and $\text{Var}[y_t] \leq \sigma^2$, where $\theta \in \mathbb{R}^n$ is an unknown parameter vector satisfying $||\theta|| \leq 1$ and $\sigma \in \mathbb{R}$ is a known constant. After observing $x_t$ and before observing $y_t$, the learning algorithm must produce an output $\hat{y}_t \in [-1, 1] \cup \{\emptyset\}$ (a prediction of $E[y_t | x_t]$). Furthermore, it should be able to provide an output $\hat{y}(x)$ for any input vector $x \in \{0, 1\}^n$.*

A key aspect of our problem that distinguishes it from other online learning models is that the algorithm is allowed to output a special value $\emptyset$ rather than make a *valid prediction* (an output other than $\emptyset$). An output of $\emptyset$ signifies that the algorithm is not sure of what to predict and therefore declines to make a prediction. The algorithm would like to minimize the number of times it predicts $\emptyset$, and, furthermore, when it does make a valid prediction the prediction must be accurate, with high probability. Next, we formalize the above intuition and define the properties of a "solution" to KLRP.

**Definition 2** *We define an* **admissible algorithm for the KWIK Linear Regression Problem** *to be one that takes two inputs $0 \leq \epsilon \leq 1$ and $0 \leq \delta < 1$ and, with probability at least $1 - \delta$, satisfies the following conditions:*

1. *Whenever the algorithm predicts $\hat{y}_t(x) \in [-1, 1]$, we have that $|\hat{y}_t(x) - \theta^T x| \leq \epsilon$.*

2. *The number of timesteps $t$ for which $\hat{y}_t(x_t) = \emptyset$ is bounded by some function $\zeta(\epsilon, \delta, n)$, polynomial in $n$, $1/\epsilon$ and $1/\delta$, called the sample complexity of the algorithm.*

## 1.1  Solution

First, we present an algorithm and then a proof that it solves KLRP. Let $X$ denote an $m \times n$ matrix whose rows we interpret as transposed input vectors. We let $X(i)$ denote the transpose of the $i$th row of $X$. Since $X^T X$ is symmetric, we can write it as

$$X^T X = U \Lambda U^T, \quad \text{(Singular Value Decomposition)} \tag{1}$$

where $U = [v_1, \ldots, v_n] \in \mathbb{R}^{n \times n}$, with $v_1, \ldots, v_n$ being a set of orthonormal eigenvectors of $X^T X$. Let the corresponding eigenvalues be $\lambda_1 \geq \lambda_2 \geq \cdots \geq \lambda_k \geq 1 > \lambda_{k+1} \geq \cdots \geq \lambda_n \geq 0$. Note that $\Lambda = \text{diag}(\lambda_1, \ldots, \lambda_n)$ is diagonal but not necessarily invertible. Now, define $\bar{U} = [v_1, \ldots, v_k] \in$

$\mathbb{R}^{n \times k}$ and $\bar{\Lambda} = \text{diag}(\lambda_1, \ldots, \lambda_k) \in \mathbb{R}^{k \times k}$. For a fixed input $x_t$ (a new input provided to the algorithm at time $t$), define

$$\bar{q} := X\bar{U}\bar{\Lambda}^{-1}\bar{U}^T x_t \in \mathbb{R}^{m \times n}, \tag{2}$$

$$\bar{v} = [0, \ldots, 0, v_{k+1}^T x_t, \ldots, v_n^T x_t]^T \in \mathbb{R}^n. \tag{3}$$

---

**Algorithm 1** KWIK Linear Regression

---

0: **Inputs:** $\alpha_1$, $\alpha_2$
1: Initialize $X = [\,]$ and $y = [\,]$.
2: **for** $t = 1, 2, 3, \cdots$ **do**
3:     Let $x_t$ denote the input at time $t$.
4:     Compute $\bar{q}$ and $\bar{v}$ using Equations 2 and 3.
5:     **if** $||\bar{q}|| \leq \alpha_1$ and $||\bar{v}|| \leq \alpha_2$ **then**
6:         Choose $\hat{\theta} \in \mathbb{R}^n$ that minimizes $\sum_i [y(i) - \bar{\theta}^T X(i)]^2$ subject to $||\bar{\theta}|| \leq 1$, where $X(i)$ is the transpose of the $i$th row of $X$ and $y(i)$ is the $i$th component of $y$.
7:         Output valid prediction $x^T \hat{\theta}$.
8:     **else**
9:         Output $\emptyset$.
10:        Receive output $y_t$.
11:        Append $x_t^T$ as a new row to the matrix $X$.
12:        Append $y_t$ as a new element to the vector $y$.
13:     **end if**
14: **end for**

---

Our algorithm for solving the KWIK Linear Regression Problem uses these quantities and is provided in pseudocode by Algorithm 1. Our first main result of the paper is the following theorem.

**Theorem 1** *With appropriate parameter settings, Algorithm 1 is an admissible algorithm for the KWIK Linear Regression Problem with a sample complexity bound of $\tilde{O}(n^3/\epsilon^4)$.*

Although the analysis of Algorithm 1 is somewhat complicated, the algorithm itself has a simple interpretation. Given a new input $x_t$, the algorithm considers making a prediction of the output $y_t$ using the norm-constrained least-squares estimator (specifically, $\hat{\theta}$ defined in line 6 of Algorithm1). The norms of the vectors $\bar{q}$ and $\bar{v}$ provide a quantitative measure of uncertainty about this estimate. When both norms are small, the estimate is trusted and a valid prediction is made. When either norm is large, the estimate is not trusted and the algorithm produces an output of $\emptyset$.

One may wonder why $\bar{q}$ and $\bar{v}$ provide a measure of uncertainty for the least-squares estimate. Consider the case when all eigenvalues of $X^T X$ are greater than 1. In this case, note that $x = X^T X (X^T X)^{-1} x = X^T \bar{q}$. Thus, $x$ can be written as a linear combination of the rows of $X$, whose coefficients make up $\bar{q}$, of previously experienced input vectors. As shown by Auer (2002), this particular linear combination minimizes $||q||$ for any linear combination $x = X^T q$. Intuitively, if the norm of $\bar{q}$ is small, then there are many previous training samples (actually, combinations of inputs) "similar" to $x$, and hence our least-squares estimate is likely to be accurate for $x$. For the case of ill-conditioned $X^T X$ (when $X^T X$ has eigenvalues close to 0), $X(X^T X)^{-1} x$ may be undefined or have a large norm. In this case, we must consider the directions corresponding to small eigenvalues separately and this consideration is dealt with by $\bar{v}$.

## 1.2 Analysis

We provide a sketch of the analysis of Algorithm 1. Please see our technical report for full details. The analysis hinges on two key lemmas that we now present.

In the following lemma, we analyze the behavior of the squared error of predictions based on an incorrect estimator $\hat{\theta} \neq \theta$ verses the squared error of using the true parameter vector $\theta$. Specifically, we show that the squared error of the former is very likely to be larger than the latter when the predictions based on $\hat{\theta}$ (of the form $\hat{\theta}^T x$ for input $x$) are highly inaccurate. The proof uses Hoeffding's bound and is omitted.

**Lemma 1** *Let $\theta \in \mathbb{R}^n$ and $\hat{\theta} \in \mathbb{R}^n$ be two fixed parameter vectors satisfying $||\theta|| \leq 1$ and $||\hat{\theta}|| \leq 1$. Suppose that $(x_1, y_1), \ldots, (x_m, y_m)$ is any sequence of samples satisfying $x_i \in \mathbb{R}^n$, $y_i \in \mathbb{R}$, $||x_i|| \leq 1$, $y_i \in [-1, 1]$, $E[y_i|x_i] = \theta^T x_i$, and $\mathrm{Var}[y_i|x_i] \leq \sigma^2$. For any $0 < \delta' < 1$ and fixed positive constant $z$, if*

$$\sum_{i=1}^{m} [(\theta - \hat{\theta})^T x_i]^2 \geq 2\sqrt{8m \ln(2/\delta)} + z, \tag{4}$$

*then*

$$\sum_{i=1}^{m} (y_i - \hat{\theta}^T x_i)^2 > \sum_{i=1}^{m} (y_i - \theta^T x_i)^2 + z \tag{5}$$

*with probability at least $1 - 2\delta'$.*

The following lemma, whose proof is fairly straight-forward and therefore omitted, relates the error of an estimate $\hat{\theta}^T x$ for a fixed input $x$ based on an inaccurate estimator $\hat{\theta}$ to the quantities $||\bar{q}||$, $||\bar{v}||$, and $\Delta_{\mathrm{E}}(\hat{\theta}) := \sqrt{\sum_{i=1}^{m} [(\theta - \hat{\theta})^T X(i)]^2}$. Recall that when $||\bar{q}||$ and $||\bar{v}||$ are both small, our algorithm becomes confident of the least-squares estimate. In precisely this case, the lemma shows that $|(\theta - \hat{\theta})^T x|$ is bounded by a quantity proportional to $\Delta_{\mathrm{E}}(\hat{\theta})$.

**Lemma 2** *Let $\theta \in \mathbb{R}^n$ and $\hat{\theta} \in \mathbb{R}^n$ be two fixed parameter vectors satisfying $||\theta|| \leq 1$ and $||\hat{\theta}|| \leq 1$. Suppose that $(x_1, y_1), \ldots, (x_m, y_m)$ is any sequence of samples satisfying $x_i \in \mathbb{R}^n$, $y_i \in \mathbb{R}$, $||x_i|| \leq 1$, $y_i \in [-1, 1]$. Let $x \in \mathbb{R}^n$ be any vector. Let $\bar{q}$ and $\bar{v}$ be defined as above. Let $\Delta_{\mathrm{E}}(\hat{\theta})$ denote the error term $\sqrt{\sum_{i=1}^{m} [(\theta - \hat{\theta})^T x_i]^2}$. We have that*

$$|(\theta - \hat{\theta})^T x| \leq ||\bar{q}||\Delta_{\mathrm{E}}(\hat{\theta}) + 2||\bar{v}||. \tag{6}$$

**Proof sketch:** (of Theorem 1)

The proof has three steps. The first is to bound the sample complexity of the algorithm (the number of times the algorithm makes a prediction of $\emptyset$), in terms of the input parameters $\alpha_1$ and $\alpha_2$. The second is to choose the parameters $\alpha_1$ and $\alpha_2$. The third is to show that, with high probability, every valid prediction made by the algorithm is accurate.

**Step 1**

We derive an upper bound $\bar{m}$ on the number of timesteps for which either $||\bar{q}|| > \alpha_1$ holds or $||\bar{v}|| > \alpha_2$ holds. Observing that the algorithm trains on only those samples experienced during pricisely these timesteps and applying Lemma 13 from the paper by Auer (2002) we have that

$$\bar{m} = O\left( \frac{n \ln(n/\alpha_1)}{\alpha_1^2} + \frac{n}{\alpha_2^2} \right). \tag{7}$$

**Step 2** We choose $\alpha_1 = C \cdot Q \ln Q$, where $C$ is a constant and $Q = \frac{\epsilon^2}{n\sqrt{\ln(1/(\epsilon\delta))}\ln(n)}$, and $\alpha_2 = \epsilon/4$.

**Step 3** Consider some fixed timestep $t$ during the execution of Algorithm 1 such that the algorithm makes a valid prediction (not $\emptyset$). Let $\hat{\theta}$ denote the solution of the norm-constrained least-squares minimization (line 6 in the pseudocode). By definition, since $\emptyset$ was not predicted, we have that $\bar{q} \leq \alpha_1$ and $\bar{v} \leq \alpha_2$. We would like to show that $|\hat{\theta}^T x - \theta^T x| \leq \epsilon$ so that Condition 1 of Definition 2 is satisfied. Suppose not, namely that $|(\hat{\theta} - \theta)^T x| > \epsilon$. Using Lemma 2, we can lower bound the quantity $\Delta_E(\hat{\theta})^2 = \sum_{i=1}^{m} [(\theta - \hat{\theta})^T X(i)]^2$, where $m$ denotes the number of rows of the matrix $X$ (equivalently, the number of samples obtained used by the algorithm for training, which we upper-bounded by $\bar{m}$), and $X(i)$ denotes the transpose of the $i$th row of $X$. Finally, we would like to apply Lemma 1 to prove that, with high probability, the squared error of $\hat{\theta}$ will be larger than the squared error of predictions based on the true parameter vector $\theta$, which contradicts the fact that $\hat{\theta}$ was chosen to minimize the term $\sum_{i=1}^{m} (y_i - \hat{\theta}^T X(i))^2$. One problem with this approach is that Lemma 1 applies to a fixed $\hat{\theta}$ and the least-squares computation of Algorithm 1 may choose any $\hat{\theta}$ in the infinite set $\{\hat{\theta} \in \mathbb{R}^n$ such that $||\hat{\theta}|| \leq 1\}$. Therefore, we use a uniform discretization to form a

finite cover of $[-1, 1]^n$ and apply the theorem to the member of the cover closest to $\hat{\theta}$. To guarantee that the total failure probability of the algorithm is at most $\delta$, we apply the union bound over all (finitely many) applications of Lemma 1. $\square$

## 1.3 Notes

In our formulation of KLRP we assumed an upper bound of $1$ on the the two-norm of the inputs $x_i$, outputs $y_i$, and the true parameter vector $\theta$. By appropriate scaling of the inputs and/or outputs, we could instead allow a larger (but still finite) bound.

Our analysis of Algorithm 1 showed that it is possible to solve KLRP with polynomial *sample complexity* (where the sample complexity is defined as the number of timesteps $t$ that the algorithm outputs $\emptyset$ for the current input $x_t$), with high probability. We note that the algorithm also has polynomial *computational complexity* per timestep, given the tractability of solving norm-constrained least-squares problems (see Chapter 12 of the book by Golub and Van Loan (1996)).

## 1.4 Related Work

Work on linear regression is abundant in the statistics community (Seber & Lee, 2003). The use of the quantities $\bar{v}$ and $\bar{q}$ to quantify the level of certainty of the linear estimator was introduced by Auer (2002). Our analysis differs from that by Auer (2002) because we do not assume that the input vectors $x_i$ are fixed ahead of time, but rather that they may be chosen in an adversarial manner. This property is especially important for the application of regression techniques to the full RL problem, rather than the Associative RL problem considered by Auer (2002). Our analysis has a similar flavor to some, but not all, parts of the analysis by Abbeel and Ng (2005). However, a crucial difference of our framework and analysis is the use of output $\emptyset$ to signify uncertainty in the current estimate, which allows for efficient exploration in the application to RL as described in the next section.

# 2 Application to Reinforcement Learning

The general reinforcement-learning (RL) problem is how to enable an agent (computer program, robot, *etc.*) to maximize an external reward signal by acting in an unknown environment. To ensure a well-defined problem, we make assumptions about the types of possible worlds. To make the problem tractable, we settle for near-optimal (rather than optimal) behavior on all but a polynomial number of timesteps, as well as a small allowable failure probability. This type of performance metric was introduced by Kakade (2003), in the vein of recent RL analyses (Kearns & Singh, 2002; Brafman & Tennenholtz, 2002).

In this section, we formalize a specific RL problem where the environment is mathematically modeled by a continuous MDP taken from a rich class of MDPs. We present an algorithm and prove that it learns efficiently within this class. The algorithm is "model-based" in the sense that it constructs an explicit MDP that it uses to reason about future actions in the true, but unknown, MDP environment. The algorithm uses, as a subroutine, any admissible algorithm for the KWIK Linear Regression Problem introduced in Section 1. Although our main result is for a specific class of continuous MDPs, albeit an interesting and previously studied one, our technique is more general and should be applicable to many other classes of MDPs as described in the conclusion.

## 2.1 Problem Formulation

The model we use is slightly modified from the model described by Abbeel and Ng (2005). The main difference is that we consider discounted rather than undiscounted MDPs and we don't require the agent to have a "reset" action that takes it to a specified start state (or distribution). Let $\mathcal{P}_S$ denote the set of all (measurable) probability distributions over the set $S$. The environment is described by a discounted MDP $M = \langle S, A, T, R, \gamma \rangle$, where $S = \mathbb{R}^{n_S}$ is the state space, $A = \mathbb{R}^{n_A}$ is the action space, $T : S \times A \rightarrow P_S$ is the unknown transition dynamics, $\gamma \in [0, 1)$ is the discount factor, and $R : S \times A \rightarrow \mathbb{R}$ is the known reward function.[1] For each timestep $t$, let $x_t \in S$ denote the current

state and $u_t \in A$ the current action. The transition dynamics $T$ satisfy

$$x_{t+1} = M\phi(x_t, u_t) + w_t, \tag{8}$$

where $x_{t+1} \in S$, $\phi(\cdot, \cdot) : \mathbb{R}^{n_S + n_A} \to \mathbb{R}^n$ is a (basis or kernel) function satisfying $||\phi(\cdot, \cdot)|| \leq 1$, and $M$ is an $n_S \times n$ matrix. We assume that the 2-norm of each row of $M$ is bounded by 1.[2] Each component of the noise term $w_t \in \mathbb{R}^{n_S}$ is chosen i.i.d. from a normal distribution with mean 0 and variance $\sigma^2$ for a known constant $\sigma$. If an MDP satisfies the above conditions we say that it is *linearly parameterized*, because the next-state $x_{t+1}$ is a linear function of the vector $\phi(x_t, u_t)$ (which describes the current state and action) plus a noise term.

We assume that the learner (also called the *agent*) receives $n_S$, $n_A$, $n$, $R$, $\phi(\cdot, \cdot)$, $\sigma$, and $\gamma$ as input, with $T$ initially being unknown. The learning problem is defined as follows. The agent always occupies a single state $s$ of the MDP $M$. The agent is given $s$ and chooses an action $a$. It then receives an *immediate reward* $r \sim \mathcal{R}(s, a)$ and is transported to a *next state* $s' \sim T(s, a)$. This procedure then repeats forever. The first state occupied by the agent may be chosen arbitrarily.

A *policy* is any strategy for choosing actions. We assume (unless noted otherwise) that rewards all lie in the interval $[0, 1]$. For any policy $\pi$, let $V_M^\pi(s)$ ($Q_M^\pi(s, a)$) denote the discounted, infinite-horizon value (action-value) function for $\pi$ in $M$ (which may be omitted from the notation) from state $s$. Specifically, let $s_t$ and $r_t$ be the $t$th encountered state and received reward, respectively, resulting from execution of policy $\pi$ in some MDP $M$ from state $s_0$. Then, $V_M^\pi(s) = E[\sum_{j=0}^\infty \gamma^j r_j | s_0 = s]$. The optimal policy is denoted $\pi^*$ and has value functions $V_M^*(s)$ and $Q_M^*(s, a)$. Note that a policy cannot have a value greater than $v_{\max} := 1/(1 - \gamma)$ by the assumption of a maximum reward of 1.

## 2.2 Algorithm

First, we discuss how to use an admissible learning algorithm for KLRP to construct an MDP model. We proceed by specifying the transition model for each of the (infinitely many) state-action pairs. Given a fixed state-action pair $(s, a)$, we need to estimate the next-state distribution of the MDP from past experience, which consists of input state-action pairs (transformed by the nonlinear function $\phi$) and output next states. For each state component $i \in \{1, \ldots, n_S\}$, we have a separate learning problem that can be solved by any instance $\mathcal{A}_i$ of an admissible KLRP algorithm.[3] If each instance makes a valid prediction (not $\emptyset$), then we simply construct an approximate next-state distribution whose $i$th component is normally distributed with variance $\sigma^2$ and whose mean is given by the prediction of $\mathcal{A}_i$ (this procedure is equivalent to constructing an approximate transition matrix $\hat{M}$ whose $i$th row is equal to the transpose of the approximate parameter vector $\hat{\theta}$ learned by $\mathcal{A}_i$).

If any instance of our KLRP algorithm predicts $\emptyset$ for state-action pair $(s, a)$, then we cannot estimate the next-state distribution. Instead, we make $s$ highly rewarding in the MDP model to encourage exploration, as done in the R-MAX algorithm (Brafman & Tennenholtz, 2002). Following the terminology introduced by Kearns and Singh (2002), we call such a state (state-action) an "unknown" state (state-action) and we ensure that the value function of our model assigns $v_{\max}$ (maximum possible) to state $s$. The standard way to satisfy this condition for finite MDPs is to make the transition function for action $a$ from state $s$ a self-loop with reward 1 (yielding a value of $v_{\max} = 1/(1 - \gamma)$ for state $s$). We can affect the exact same result in a continuous MDP by adding a component to each state vector $s$ and to each vector $\phi(s, a)$ for every state-action pair $(s, a)$. If $(s, a)$ is "unknown" we set the value of the additional components (of $\phi(s, a)$ and $s$) to 1, otherwise we set it to 0. We add an additional row and column to $M$ that preserves this extra component (during the transformation from $\phi(s, a)$ to the next state $s'$) and otherwise doesn't change the next-state distribution. Finally, we give a reward of 1 to any unknown state, leaving rewards for the known states unchanged. Pseudocode for the resulting KWIK-RMAX algorithm is provided in Algorithm 2.

**Theorem 2** *For any $\epsilon$ and $\delta$, the* KWIK-RMAX *algorithm executes an $\epsilon$-optimal policy on at most a polynomial (in $n$, $n_S$, $1/\epsilon$, $1/\delta$, and $1/(1-\gamma)$) number of steps, with probability at least $1 - \delta$.*

**Algorithm 2** KWIK-RMAX Algorithm

---

0: **Inputs:** $n_S$, $n_A$, $n$, $R$, $\phi(\cdot, \cdot)$, $\sigma$, $\gamma$, $\epsilon$, $\delta$, and admissible learning algorithm $\mathcal{M}odel\mathcal{L}earn$.

1: **for all** state components $i \in \{1, \ldots, n_S\}$ **do**

2:     Initialize a new instantiation of $\mathcal{M}odel\mathcal{L}earn$, denoted $\mathcal{A}_i$, with inputs $C\frac{\epsilon(1-\gamma)^2}{2\sqrt{n}}$ and $\delta/n_S$, for inputs $\epsilon$ and $\delta$, respectively, in Definition 2, and where $C$ is some constant determined by the analysis.

3: **end for**

4: Initialize an MDP $\mathcal{M}odel$ with state space $S$, action space $A$, reward function $R$, discount factor $\gamma$ and transition function specified by $\mathcal{A}_i$ for $i \in \{1, \ldots, n_S\}$ as described above.

5: **for** $t = 1, 2, 3, \cdots$ **do**

6:     Let $s$ denote the state at time $t$.

7:     Choose action $a := \hat{\pi}^*(s)$ where $\hat{\pi}^*$ is the optimal policy of the MDP $\mathcal{M}odel$.

8:     Let $s'$ be the next state after executing action $a$.

9:     **for all** factors $i \in \{1, \ldots, n\}$ **do**

10:         Present input-output pair $(\phi(s, a), s'(i))$ to $\mathcal{A}_{i,a}$.

11:     **end for**

12:     Update MDP $\mathcal{M}odel$.

13: **end for**

---

### 2.3 Analysis

**Proof sketch:** (of Theorem 2)

It can be shown that, with high probability, policy $\hat{\pi}^*$ is either an $\epsilon$-optimal policy ($V^{\hat{\pi}^*}(s) \geq V^*(s) - \epsilon$) or it is very likely to lead to an unknown state. However, the number of times the latter event can occur is bounded by the maximum number of times the instances $\mathcal{A}_i$ can predict $\emptyset$, which is polynomial in the relevant parameters. $\square$

### 2.4 The Planning Assumption

We have shown that the KWIK-RMAX Algorithm acts near-optimally on all but a small (polynomial) number of timesteps, with high probability. Unfortunately, to do so, the algorithm must solve its internal MDP model completely and exactly. It is easy to extend the analysis to allow $\epsilon$-approximate solution. However, it is not clear whether even this approximate computation can be done efficiently. In any case, discretization of the state space can be used, which yields computational complexity that is exponential in the number of (state and action) dimensions of the problem, similar to the work of Chow and Tsitsiklis (1991). Alternatively, sparse sampling can be used, whose complexity has no dependence on the size of the state space but depends exponentially on the time horizon ($\approx 1/(1 - \gamma)$) (Kearns et al., 1999). Practically, there are many promising techniques that make use of value-function approximation for fast and efficient solution (planning) of MDPs (Sutton & Barto, 1998). Nevertheless, it remains future work to fully analyze the complexity of planning.

### 2.5 Related Work

The general exploration problem in continuous state spaces was considered by Kakade et al. (2003), and at a high level our approach to exploration is similar in spirit. However, a direct application of Kakade et al.'s (2003) algorithm to linearly-parameterized MDPs results in an algorithm whose sample complexity scales exponentially, rather than polynomially, with the state-space dimension. That is because the analysis uses a factor of the size of the "cover" of the metric space. Reinforcement learning in continuous MDPs with linear dynamics was studied by Fiechter (1997). However, an exact linear relationship between the current state and next state is required for this analysis to go through, while we allow the current state to be transformed (for instance, adding non-linear state features) through non-linear function $\phi$. Furthermore, Fiechter's algorithm relied on the existence of a "reset" action and a specific form of reward function. These assumptions admit a solution that follows a fixed policy and doesn't depend on the actual history of the agent or the underlying MDP. The model that we consider, linearly parameterized MDPs, is taken directly from the work by Abbeel and Ng (2005), where it was justified in part by an application to robotic helicopter flight. In

that work, a provably efficient algorithm was developed in the *apprenticeship RL* setting. In this setting, the algorithm is given limited access (polynomial number of calls) to a fixed policy (called the *teacher's policy*). With high probably, a policy is learned that is nearly as good as the teacher's policy. Although this framework is interesting and perhaps more useful for certain applications (such as helicopter flying), it requires *a priori* expert knowledge (to construct the teacher) and alleviates the problem of exploration altogether. In addition, Abbeel and Ng's (2005) algorithm also relies heavily on a reset assumption, while ours does not.

## Conclusion

We have provided a provably efficient RL algorithm that learns a very rich and important class of MDPs with continuous state and action spaces. Yet, many real-world MDPs do not satisfy the linearity assumption, a concern we now address. Our RL algorithm utilized a specific online linear regression algorithm. We have identified certain interesting and general properties (see Definition 2) of this particular algorithm that support online exploration. These properties are meaningful without the linearity assumption and should be useful for development of new algorithms for different modeling assumptions. Our real goal of the paper is to work towards developing a general technique for applying regression algorithms (as black boxes) to model-based reinforcement-learning algorithms in a robust and formally justified way. We believe the approach used with linear regression can be repeated for other important classes, but we leave the details as interesting future work.

## Acknowledgements

We thank NSF and DARPA IPTO for support.

## Footnotes

[1] All of our results can easily be extended to the case of an unknown reward function with a suitable linearity assumption.

[2]The algorithm can be modified to deal with bounds (on the norms of the rows of $M$) that are larger than one.

[3]One minor technical detail is that our KLRP setting requires bounded outputs (see Definition 1) while our application to MBRL requires dealing with normal, and hence unbounded outputs. This is easily dealt with by ignoring any extremely large (or small) outputs and showing that the resulting norm of the *truncated normal distribution* learned by the each instance $\mathcal{A}_i$ is very close to the norm of the untruncated distribution.

## References

Abbeel, P., & Ng, A. Y. (2005). Exploration and apprenticeship learning in reinforcement learning. *ICML '05: Proceedings of the 22nd international conference on Machine learning* (pp. 1–8). New York, NY, USA: ACM Press.

Auer, P. (2002). Using confidence bounds for exploitation-exploration trade-offs. *Journal of Machine Learning Research*, *3*, 397–422.

Brafman, R. I., & Tennenholtz, M. (2002). R-MAX—a general polynomial time algorithm for near-optimal reinforcement learning. *Journal of Machine Learning Research*, *3*, 213–231.

Chow, C.-S., & Tsitsiklis, J. N. (1991). An optimal one-way multigrid algorithmfor discrete time stochastic control. *IEEE Transactions on Automatic Control*, *36*, 898–914.

Fiechter, C.-N. (1997). PAC adaptive control of linear systems. *Tenth Annual Conference on Computational Learning Theory (COLT)* (pp. 72–80).

Golub, G. H., & Van Loan, C. F. (1996). *Matrix computations*. Baltimore, Maryland: The Johns Hopkins University Press. 3rd edition.

Kakade, S. M. (2003). *On the sample complexity of reinforcement learning*. Doctoral dissertation, Gatsby Computational Neuroscience Unit, University College London.

Kakade, S. M. K., Kearns, M. J., & Langford, J. C. (2003). Exploration in metric state spaces. *Proceedings of the 20th International Conference on Machine Learning (ICML-03)*.

Kearns, M., Mansour, Y., & Ng, A. Y. (1999). A sparse sampling algorithm for near-optimal planning in large Markov decision processes. *Proceedings of the Sixteenth International Joint Conference on Artificial Intelligence (IJCAI-99)* (pp. 1324–1331).

Kearns, M. J., & Singh, S. P. (2002). Near-optimal reinforcement learning in polynomial time. *Machine Learning*, *49*, 209–232.

Ng, A. Y., Kim, H. J., Jordan, M. I., & Sastry, S. (2003). Autonomous helicopter flight via reinforcement learning. *Advances in Neural Information Processing Systems 16 (NIPS-03)*.

Seber, G. A. F., & Lee, A. J. (2003). *Linear regression analysis*. Wiley-Interscience.

Sutton, R. S., & Barto, A. G. (1998). *Reinforcement learning: An introduction*. The MIT Press.

Tesauro, G. (1994). TD-Gammon, a self-teaching backgammon program, achieves master-level play. *Neural Computation*, *6*, 215–219.

